# Mutual Boosting for Contextual Inference

**Michael Fink**
Center for Neural Computation
Hebrew University of Jerusalem
Jerusalem, Israel 91904
*fink@huji.ac.il*

**Pietro Perona**
Electrical Engineering Department
California Institute of Technology
Pasadena, CA 91125
*perona@vision.caltech.edu*

## Abstract

Mutual Boosting is a method aimed at incorporating contextual information to augment object detection. When multiple detectors of objects and parts are trained in parallel using AdaBoost [1], object detectors might use the remaining intermediate detectors to enrich the weak learner set. This method generalizes the efficient features suggested by Viola and Jones [2] thus enabling information inference between parts and objects in a compositional hierarchy. In our experiments eye-, nose-, mouth- and face detectors are trained using the Mutual Boosting framework. Results show that the method outperforms applications overlooking contextual information. We suggest that achieving contextual integration is a step toward human-like detection capabilities.

## 1  Introduction

Classification of multiple objects in complex scenes is one of the next challenges facing the machine learning and computer vision communities. Although, real-time detection of single object classes has been recently demonstrated [2], naïve duplication of these detectors to the multiclass case would be unfeasible. Our goal is to propose an efficient method for detection of multiple objects in natural scenes.

Hand-in-hand with the challenges entailing multiclass detection, some distinct advantages emerge as well. Knowledge on position of several objects might shed light on the entire scene (Figure 1). Detection systems that do not exploit the information provided by objects on the neighboring scene will be suboptimal.

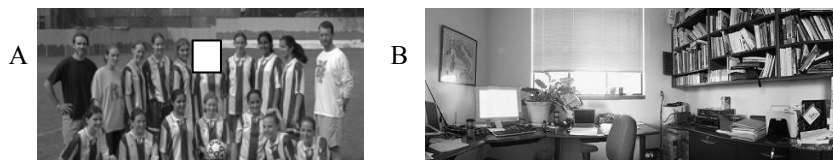

Figure 1: Contextual spatial relationships assist detection **A.** in absence of facial components (whitened blocking box) faces can be detected by context (alignment of neighboring faces). **B.** keyboards can be detected when they appear under monitors.

Many human and computer vision models postulate explicitly or implicitly that vision follows a compositional hierarchy. Grounded features (that are innate/hardwired and are available prior to learning) are used to detect salient parts, these parts in turn enable detection of complex objects [3, 4], and finally objects are used to recognize the semantics of the entire scene. Yet, a more accurate assessment of human performance reveals that the visual system often violates this strictly hierarchical structure in two ways. First, part and whole detection are often evidently interacting [5, 6]. Second, several layers of the hierarchy are occasionally bypassed to enable swift direct detection. This phenomenon is demonstrated by gist recognition experiments where the semantic classification of an entire scene is performed using only minimal low level feature information [7].

The insights emerging from observing human perception were adopted by the object detection community. Many object detection algorithms bypass stages of a strict compositional hierarchy. The Viola & Jones (VJ) detector [2] is able to perform robust online face detection by directly agglomerating very low-level features (rectangle contrasts), without explicitly referring to facial parts. Gist detection from low-level spatial frequencies was demonstrated by Oliva and Torralba [8]. Recurrent optimization of parts and object constellation is also common in modern detection schemes [9]. Although Latent Semantic Analysis (making use of object co-occurrence information) has been adapted to images [10], the existing state of object detection methods is still far from unifying all the sources of visual contextual information integrated by the human perceptual system. Tackling the context integration problem and achieving robust multiclass object detection is a vital step for applications like image-content database indexing and autonomous robot navigation.

We will propose a method termed *Mutual Boosting* to incorporate contextual information for object detection. Section 2 will start by posing the multiclass detection problem from labeled images. In Section 3 we characterize the feature sets implemented by Mutual Boosting and define an object's contextual neighborhood. Section 4 presents the Mutual Boosting framework aimed at integrating contextual information and inspired by the recurrent inferences dominating the human perceptual system. An application of the Mutual Boosting framework to facial component detection is presented in Section 5. We conclude with a discussion on the scope and limitations of the proposed framework.

## 2   Problem setting and basic notation

Suppose we wish to detect multiple objects in natural scenes, and that these scenes are characterized by certain mutual positions between the composing objects. Could we make use of these objects' contextual relations to improve detection? Perceptual context might include multiple sources of information: information originating from the presence of existing parts, information derived from other objects in the perceptual vicinity and finally general visual knowledge on the scene. In order to incorporate these various sources of visual contextual information Mutual Boosting will treat parts, objects and scenes identically. We will therefore use the term object as a general term while referring to any entity in the compositional hierarchy.

Let M denote the cardinality of the object set we wish to detect in natural scenes. Our goal is to optimize detection by exploiting contextual information while maintaining detection time comparable to M individual detectors trained without such information. We define the goal of the multiclass detection algorithm as generating M intensity maps $H^{m=1,...,M}$ indicating the likelihood of object m appearing at different positions in a target image.

We will use the following notation (Figure 2):

- $H^{0+}/H^{0-}$: raw image input with/without the trained objects ($A_1$ & $A_2$)

- $C^{m[i]}$: labeled position of instance i of object m in image $H^{0+}$

- $H^m$: intensity map output indicating the likelihood of object m appearing in different positions in the image $H^0$ (**B**)

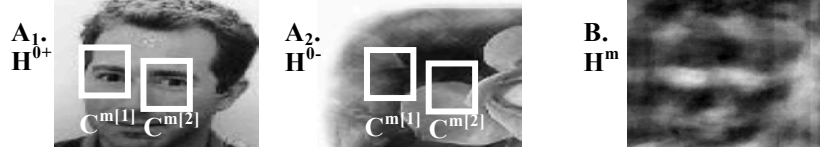

Figure 2: $A_1$ & $A_2$. **Input:** position of positive and negative examples of eyes in natural images. **B. Output:** Eye intensity (eyeness) detection map of image $H^{0+}$

## 3  Feature set and contextual window generalizations

The VJ method for real-time object-detection included three basic innovations. First, they presented the *rectangle contrast-features*, features that are evaluated efficiently, using an integral-image. Second, VJ introduced AdaBoost [1] to object detection using rectangle features as weak learners. Finally a cascade method was developed to chain a sequence of increasingly complex AdaBoost learners to enable rapid filtering of non-relevant sections in the target image. The resulting cascade of AdaBoost face detectors achieves a 15 frame per second detection speed, with 90% detection rate and $2\text{x}10^{-6}$ false alarms. This detection speed is currently unmatched. In order to maintain efficient detection and in order to benchmark the performance of Mutual Boosting we will adopt the rectangle contrast feature framework suggested by VJ.

It should be noted that the grayscale rectangle features could be naturally extended to any image channel that preserves the semantics of summation. A diversified feature set (including color features, texture features, etc.) might saturate later than a homogeneous channel feature set. By making use of features that capture the object regularities well, one can improve performance or reduce detection time.

VJ extract training windows that capture the exact area of the training faces. We term this the *local window* approach. A second approach, in line with our attempt to incorporate information from neighboring parts or objects, would be to make use of training windows that capture wide regions around the object (Figure 3)[1].

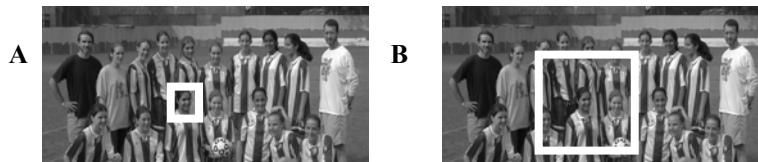

Figure 3: A local window (VJ) and a contextual window that captures relative position information from objects or parts around and within the detected object.

The *contextual neighborhood* approach contributes to detection when the applied channels require a wide contextual range as will be demonstrated in the Mutual Boosting scheme presented in the following section[2].

## 4   Mutual Boosting

The AdaBoost algorithm maintains a clear distinction between the boosting level and the weak-learner training level. The basic insight guiding the Mutual Boosting method reexamines this distinction, stipulating that when multiple objects and parts are trained simultaneously using AdaBoost; any object detector might combine the previously evolving intermediate detectors to generate new weak learners. In order to elaborate this insight it should first be noted that while training a strong learner using 100 iterations of AdaBoost (abbreviated AB100) one could calculate an intermediate strong learner at each step on the way (AB2 - AB99). To apply this observation for our multiclass detection problem we simultaneously train M object detectors. At each boosting iteration t the M detectors ($AB^m_{t-1}$) emerging at the previous stage t-1, are used to filter positive and negative[3] training images, thus producing intermediate m-detection maps $H^m_{t-1}$ (likelihood of object m in the images[4]). Next, the Mutual Boosting stage takes place and all the existing $\mathbf{H^m_{t-1}}$ **maps are used as additional channels** out of which new contrast features are selected. This process gradually enriches the initial grounded features with composite contextual features. The composite features are searched on a PxP wide contextual neighborhood region rather than the PxP local window (Figure 3).

Following a dynamic programming approach in training and detection, $H^{m=1,..,M}$ detection maps are constantly maintained and updated so that the recalculation of $H^m_t$ only requires the last chosen weak learner $WL^{mn^*}_t$ to be evaluated on channel $H^{n^*}_{t-1}$ of the training image (Figure 4). This evaluation produces a binary detection layer that will be weighted by the AdaBoost weak-learner weighting scheme and added to the previous stage map[5].

Although Mutual Boosting examines a larger feature set during training, an iteration of Mutual Boosting detection of M objects is as time-consuming as performing an AdaBoost detection iteration for M individual objects. The advantage of Mutual Boosting emerges from introducing highly informative feature sets that can enhance detection or require fewer boosting iterations. While most object detection applications extract a local window containing the object information and discard the remaining image (including the object positional information). Mutual Boosting processes the entire image during training and detection and makes constant use of the information characterizing objects' relative-position in the training images.

As we have previously stated, the detected objects might be in various levels of a compositional hierarchy (e.g. complex objects or parts of other objects). Nevertheless, Mutual Boosting provides a similar treatment to objects, parts and scenes enabling any compositional structure of the data to naturally emerge. We will term any contextual reference that is not directly grounded to the basic features, as a *cross referencing* of objects[6].

| | |
|---|---|
| **Input** | $H^{0+/0-}$ positive / negative raw images |
| | $C^{m[i]}$ position of instance $i$ of object $m=1,..,M$ in image $H^{0+}$ |
| **Initialization** | initialize boosting-weights of instances $i$ of object $m$ to **1** |
| | initialize detection maps $H^{m+}_0 / H^{m-}_0$ to **0** |

**For** $t=1,...,T$

    **For** $m=1,..,M$ *and* $n=0,..,M$
      **(A)** cutout & downscale local ($n=0$) or contextual ($n>0$) windows ($WIN^m$)
         of instances $i$ of object $m$ (at $C^{m[i]}$), from all existing images $H^n_{t-1}$

    **For** $m=1,..,M$
      normalize boosting-weights of object $m$ instances [1]
      **(B$_{1\&2}$)** select map $H^{n*}_{t-1}$ and weak learner $WL^{mn*}$ that minimize error on $WIN^m$
         decrease boosting-weights of instances that $WL^{mn*}$ labeled correctly [1]
      **(C)** DetectionLayer$^{mn*} \leftarrow WL^{mn*}(H^{n*}_{t-1})$
         calculate $\alpha^m_t$ the weak learner contribution factor from the empirical error [1]
      **(D)** update $m$-detection map $H^m_t \leftarrow H^m_{t-1} + \alpha^m_t$ DetectionLayer$^{mn*}$

**Return** strong learner $AB^m_T$ including $WL^{mn*}_{1,...,T}$ and $\alpha^m_{1,...,T}$ $(m=1,..,M)$

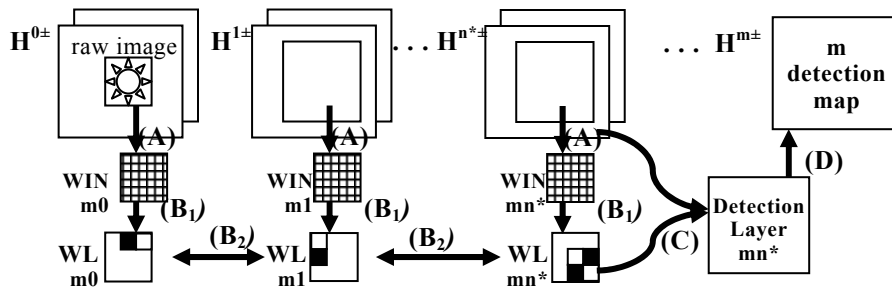

Figure 4: Mutual Boosting Diagram & Pseudo code. Each raw image $H^0$ is analyzed by M object detection maps $H^{m=1,..,M}$, updated by iterating through four steps: **(A)** cutout & downscale from existing maps $H^{n=0,...,M}_{t-1}$ a local ($n=0$) or contextual ($n>0$) PxP window containing a neighborhood of object $m$ **(B$_{1\&2}$)** select best performing map $H^{n*}$ and weak learner $WL^{mn*}$ that optimize object $m$ detection **(C)** run $WL^{mn*}$ on $H^{n*}$ map to generate a new binary $m$-detection layer **(D)** add $m$-detection layer to existing detection map $H^m$. [1] Standard AdaBoost stages are not elaborated

To maintain local and global natural scene statistics, negative training examples are generated by pairing each image with an image of equal size that does not contain the target objects and by centering the local and contextual windows of the positive and negative examples on the object positions in the positive images (see Figure 2). By using parallel boosting and efficient rectangle contrast features, Mutual Boosting is capable of incorporating many information inferences (references in Figure 5):

- Features could be used to directly detect parts and objects (**A & B**)
- Objects could be used to detect other (or identical) objects in the image (**C**)
- Parts could be used to detect other (or identical) nearby parts (**D & E**)
- Parts could be used to detect objects (**F**)
- Objects could be used to detect parts

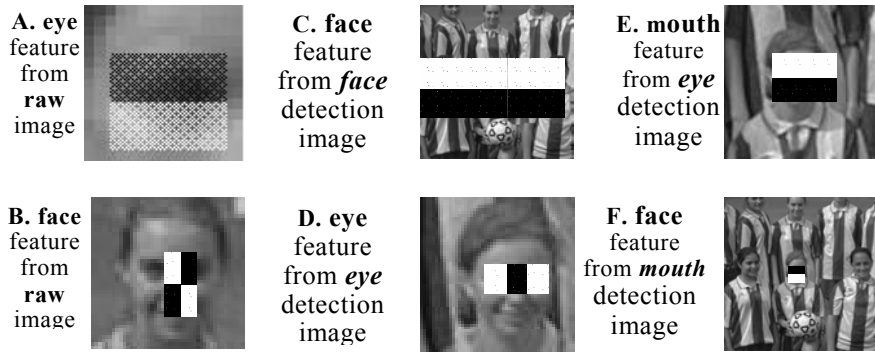

**A. eye** feature from **raw** image

**B. face** feature from **raw** image

**C. face** feature from *face* detection image

**D. eye** feature from *eye* detection image

**E. mouth** feature from *eye* detection image

**F. face** feature from *mouth* detection image

Figure 5: **A-E.** Emerging features of eyes, mouths and faces (presented on windows of raw images for legibility). The windows' scale is defined by the detected object size and by the map mode (local or contextual). **C.** faces are detected using face detection maps $H^{Face}$, exploiting the fact that faces tend to be horizontally aligned.

## 5   Experiments

In order to test the contribution of the Mutual Boosting process we focused on detection of objects in what we term a *face-scene* (right eye, left eye, nose, mouth and face). We chose to perform contextual detection in the face-scene for two main reasons. First as detailed in Figure 5, face scenes demonstrate a range of potential part and object cross references. Second, faces have been the focus of object detection research for many years, thus enabling a systematic result comparison. Experiment 1 was aimed at comparing the performance of Mutual Boosting to that of naïve independently trained object detectors using local windows.

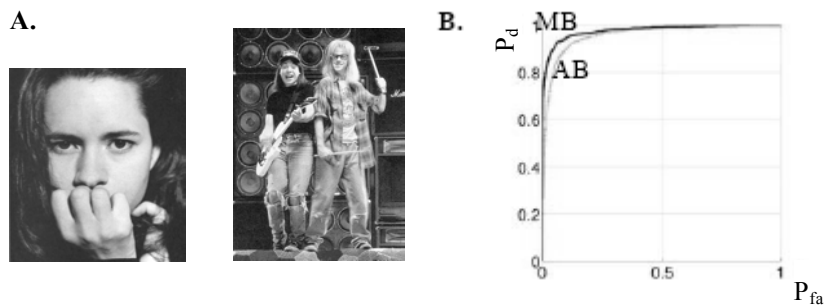

Figure 6: **A.** Two examples of the CMU/MIT face database. **B.** Mutual Boosting and AdaBoost ROCs on the CMU/MIT face database.

Face-scene images were downloaded from the web and manually labeled[7]. Training relied on 450 positive and negative examples (~4% of the images used by VJ). 400 iterations of local window AdaBoost and contextual window Mutual Boosting were performed on the same image set. Contextual windows encompassed a region five times larger in width and height than the local windows[8] (see Figure 3).

Test image detection maps emerge from iteratively summing T m-detection layers (Mutual Boosting stages **C&D**). ROC performance on the CMU/MIT face database (see sample images in Figure 6A) was assessed using a threshold on position $C^{m[i]}$ that best discriminated the final positive and negative detection maps $H^{m+/-}_T$. Figure 6B demonstrates the superiority of Mutual Boosting to grounded feature AdaBoost.

Our second experiment was aimed at assessing the performance of Mutual Boosting as we change the detected configurations' variance. Assuming normal distribution of face configurations we estimated (from our existing labeled set) the spatial covariance between four facial components (noses, mouths and both eyes). We then modified the covariance matrix, multiplying it by 0.25, 1 or 4 and generated 100 artificial configurations by positioning four contrasting rectangles in the estimated position of facial components. Although both Mutual Boosting and AdaBoost performance degraded as the configuration variance increased, the advantage of Mutual Boosting persists both in rigid and in varying configurations[9] (Figure 7).

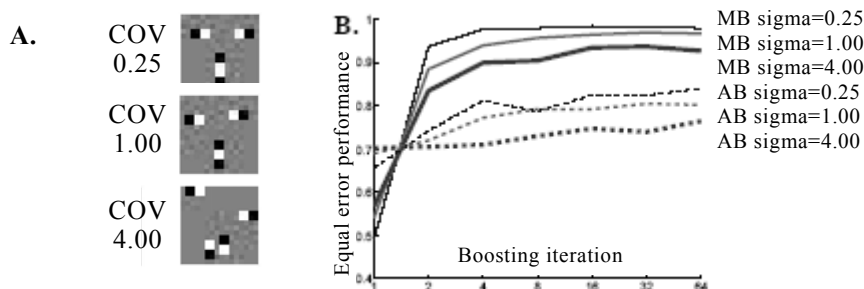

Figure 7: **A.** Artificial face configurations with increasing covariance **B.** MB and AB Equal error rate performance on configurations with varying covariance as a function of boosting iterations.

# 6 Discussion

While evaluating the performance of Mutual Boosting it should be emphasized that we did not implement the VJ cascade approach; therefore we only attempt to demonstrate that the power of a single AdaBoost learner could be augmented by Mutual Boosting. The VJ detector is rescaled in order to perform efficient detection of objects in multiple scales. For simplicity, scale of neighboring objects and parts was assumed to be fixed so that a similar detector-rescaling approach could be followed. This assumption holds well for face-scenes, but if neighboring objects may vary in scale a single m-detection map will not suffice. However, by transforming each m-detection image to an m-detection cube, (having scale as the third dimension) multi-scale context detection could be achieved[10]. The dynamic programming characteristic of Mutual Boosting (simply reusing the multiple position and scale detections already performed by VJ) will ensure that the running time of varying scale context will only be doubled. It should be noted that the face-scene is highly structured and therefore it is a good candidate for demonstrating

Mutual Boosting; however as suggested by Figure 7B Mutual Boosting can handle highly varying configurations and the proposed method needs no modification when applied to other scenes, like the office scene in Figure 1[11]. Notice that Mutual Boosting does not require a-priori knowledge of the compositional structure but rather permits structure to naturally emerge in the cross referencing pattern (see examples in Figure 5).

Mutual Boosting could be enhanced by unifying the selection of weak-learners rather than selecting an individual weak learner for each object detector. Unified selection is aimed at choosing weak learners that maximize the entire object set detection rate, thus maximizing feature reuse [11]. This approach is optimal when many objects with common characteristics are trained.

Is Mutual Boosting specific for image object detection? Indeed it requires labeled input of multiple objects in a scene supplying a local description of the objects as well as information on their contextual mutual positioning. But these criterions are shared by other complex "scenes". DNA sequences include multiple objects (Genes) in mutual positions, and therefore might be handled by a variant of Mutual Boosting. The remarkable success of the VJ method stems from abandoning the use of highly custom-tailored complex features in favor of numerous simple ones. Mutual Boosting combines parallel boosting, with a similar feature approach to efficiently incorporate contextual information. We suggest that achieving wide contextual integration is one step towards human-like object detection capabilities.

## Footnotes

[1] Contextual neighborhoods emerge by downscaling **larger regions** in the original image to a PxP resolution window.

[2] The most efficient size of the contextual neighborhoods might vary, from the immediate to the entire image, and therefore should be empirically learned.

[3] Images without target objects (see experimental section below)

[4] Unlike the weak learners, the intermediate strong learners do not apply a threshold

[5] In order to optimize the number of detection map integral image recalculations these maps might be updated every k (e.g. 50) iterations rather than at each iteration.

[6] Scenes can be crossed referenced as well if scene labels are available (office/lab etc.).

[7] By following CMU database conventions (R-eye, L-eye, Nose & Mouth positions) we derive both the local window position and the relative position of objects in the image

[8] Local windows were created by downscaling objects to 25x25 grids

[9] In this experiment the resolution of the MB windows (and the number of training features) was decreased so that information derived from the higher resolution of the parts would be ruled out as an explaining factor for the Mutual Boosting advantage. This procedure explains the superior AdaBoost performance in the first boosting iteration.

[10] By using an integral cube, calculating the sum of a cube feature (of any size) requires 8 access operations (only double than the 4 operations required in the integral image case).

[11] MB is currently aimed at detecting objects in office-scenes (Caltech 360° office DB)

## References

[1] Freund, Y. and Schapire, R. E. (1997) A Decision-Theoretic Generalization of On-Line Learning and an Application to Boosting. JCSS 55(1): 119-139

[2] Viola, V. P. and Jones M. (2001) Robust real-time object detection. IEEE ICCV Workshop on Stat. and Comp. Theories of Vision , Vancouver, Canada, July 13 2001

[3] Tanaka, K., Saito, H., Fukada, Y. and Moriya, M. (1991) Coding visual images of objects in the inferotemporal cortex of the macaque monkey. J. Neurophys. 66:170-189

[4] Biederman, I. (1987). Recognition-by-components: A theory of human image understanding. Psychological Review, 94, 115±147.

[5] Navon, D. (1977). Forest before trees: The precedence of global features in visual perception. Cog. Psych. 9, 353-383.

[6] Biederman, I., Mezzanotte, R. J., & Rabinowitz, J. C. (1982). Scene perception: Detecting the judging objects undergoing relational violations. Cog. Psych. 14, 143±177

[7] Biederman, I. (1981). On the semantics of a glance at a scene. In M. Kubovy, & J. R. Pomerantz, Perceptual organization (pp. 213±253). Hillsdale, NJ: Erlbaum.

[8] Oliva, A., Torralba, A. B. (2002) Scene-Centered Description from Spatial Envelope Properties. Biologically Motivated Computer Vision 2002: 263-272

[9] Weber, M., Welling, M., & Perona, P. (2000) Unsupervised Learning of Models for Recognition. ECCV (1) 2000: 18-32

[10] Barnard K. and Forsyth D. (2001) Learning the semantics of words and pictures. In IEEE ICCV, volume 2, pages 408--415, Vancouver, Canada, July 2001

[11] Schapire, R. E. and Singer. Y. (2000) Boostexter: A boosting-based system for text categorization. Machine Learning, 39(2-3):135--168, May/June 2000.

